# Stratification Learning: Detecting Mixed Density and Dimensionality in High Dimensional Point Clouds

**Gloria Haro, Gregory Randall, and Guillermo Sapiro**
IMA and Electrical and Computer Engineering
University of Minnesota, Minneapolis, MN 55455
`haro@ima.umn.edu,randall@fing.edu.uy,guille@umn.edu`

## Abstract

The study of point cloud data sampled from a stratification, a collection of manifolds with possible different dimensions, is pursued in this paper. We present a technique for simultaneously soft clustering and estimating the mixed dimensionality and density of such structures. The framework is based on a maximum likelihood estimation of a Poisson mixture model. The presentation of the approach is completed with artificial and real examples demonstrating the importance of extending manifold learning to stratification learning.

## 1 Introduction

Data in high dimensions is becoming ubiquitous, from image analysis and finances to computational biology and neuroscience. This data is often given or represented as samples embedded in a high dimensional Euclidean space, *point cloud data*, though it is assumed to belong to lower dimensional manifolds. Thus, in recent years, there have been significant efforts in the development of methods to analyze these point clouds and their underlying manifolds. These include numerous techniques for the estimation of the intrinsic dimension of the data and also its projection onto lower dimensional representations. These disciplines are often called *manifold learning* and *dimensionality reduction*. A few examples include [2, 3, 4, 9, 10, 11, 12, 16].

The vast majority of the manifold learning and dimensionality reduction techniques developed in the literature assume, either explicitly or implicitly, that the given point cloud are samples of a unique manifold. It is very easy to realize that a significant part of the interesting data has mixed dimensionality and complexity. The work here presented deals with this more general case, where there are different dimensionalities/complexities present in the point cloud data. That is, we have samples not of a manifold but of a *stratification*. The main aim is to cluster the data according to the complexity (dimensionality) of the underlying possible multiple manifolds. Such clustering can be used both to better understand the varying dimensionality and complexity of the data, e.g., states in neural recordings or different human activities for video analysis, or as a pre-processing step for the above mentioned manifold learning and dimensionality reduction techniques.

This clustering-by-dimensionality task has been recently explored in a handful of works. Barbará and Chen, [1], proposed a hard clustering technique based on the fractal dimension (box-counting). Starting from an initial clustering, they incrementally add points into the cluster for which the change in the fractal dimension after adding the point is the lowest. They also find the number of clusters and the intrinsic dimension of the underlying manifolds. Gionis *et al.*, [7], use local growth curves to estimate the local correlation dimension and density for each point. The new two-dimensional representation of the data is clustered using standard techniques. Souvenir and Pless, [14], use an Expectation Maximization (EM) type of technique, combined with weighted geodesic multidimensional scaling. The weights measure how well each point fits the underlying manifold defined by the current set of points in the cluster. After clustering, each cluster dimensionality is estimated

following [10]. Huang *et al.*, [8], cluster linear subspaces with an algebraic geometric method based on polynomial differentiation and a Generalized PCA. They search for the best combination of linear subspaces that explains the data, and find the number of linear subspaces and their intrinsic dimension. The work of Mordohai and Medioni, [11], estimates the local dimension using tensor voting.

These recent works have clearly shown the necessity to go beyond manifold learning, into "stratification learning." In our work, we do not assume linear subspaces, and we simultaneously estimate the soft clustering and the intrinsic dimension and density of the clusters. This collection of attributes is not shared by any of the pioneering works just described. Our approach is an extension of the Levina and Bickel's local dimension estimator [10]. They proposed to compute the intrinsic dimension at each point using a Maximum Likelihood (ML) estimator based on a Poisson distribution. The local estimators are then averaged, under the assumption of a single uniform manifold. We propose to compute a ML on the whole point cloud data at the same time (and not one for each point independently), and use a Poisson mixture model, which permits to have different classes, each one with their own dimension and sampling density. This technique automatically gives a soft clustering according to dimensionality and density, with an estimation of both quantities for each class. Our approach assumes that the number of classes is given, but we are discovering the actual number of underlying manifolds. If we search for a larger than needed number of classes, we obtain some classes with the same dimensionality and density or some classes with very few representatives, as shown in the examples later presented.

The remainder of this paper is organized as follows: In Section 2 we review the method proposed by Levina and Bickel, [10], which gives a local estimation of the intrinsic dimension and has inspired our work. In Section 3 we present our core contribution of simultaneous soft clustering and dimensionality and density estimation. We present experiments with synthetic and real data in Section 4, and finally, some conclusions are presented in Section 5.

## 2 Local intrinsic dimension estimation

Levina and Bickel (LB), [10], proposed a geometric and probabilistic method which estimates the local dimension (and density) of a point cloud data[1]. This is the approach we here extend, which is based on the idea that if we sample an $m$-dimensional manifold with $T$ points, the proportion of points that fall into a ball around a point $x_t$ is $\frac{k}{T} \approx f(x_t)V(m)R_k(x_t)^m$, where the given point cloud, embedded in high dimensions $D$, is $X = \{x_t \in \mathbb{R}^D; t = 1, \ldots, T\}$, $k$ is the number of points inside the ball, $f(x_t)$ is the local sampling density at point $x_t$, $V(m)$ is the volume of the unit sphere in $\mathbb{R}^m$, and $R_k(x_t)$ is the Euclidean distance from $x_t$ to its $k$-th nearest neighbor (kNN). Then, they consider the inhomogeneous process $N(R, x_t)$, which counts the number of points falling into a small $D$-dimensional sphere $B(R, x_t)$ of radius $R$ centered at $x_t$. This is a binomial process, and some assumptions need to be done to proceed. First, if $T \to \infty$, $k \to \infty$, and $k/T \to 0$, then we can approximate the binomial process by a Poisson process. Second, the density $f(x_t)$ is constant inside the sphere, a valid assumption for small $R$. With these assumptions, the rate $\lambda$ of the counting process $N(R, x_t)$ can be written as $\lambda(R, x_t) = f(x_t)V(m)mR^{m-1}$. The log-likelihood of the process $N(R, x_t)$ is then given by

$$L(m(x_t), \theta(x_t)) = \int_0^R \log \lambda(r, x_t)dN(r, x_t) - \int_0^R \lambda(r, x_t)dr,$$

where $\theta(x_t) := \log f(x_t)$ is the density parameter and the first integral is a Riemann-Stieltjes integral [13]. The maximum likelihood estimators satisfy $\partial L/\partial \theta = 0$ and $\partial L/\partial m = 0$, leading to a computation for the local dimension at point $x_t$, $m(x_t)$, depending on all the neighbors within a distance $R$ from $x_t$ [10]. In practice, it is more convenient to compute a fixed amount $k$ of nearest neighbors. Thus, the local dimension at point $x_t$ is $m(x_t) = \left[ \frac{1}{k-2} \sum_{j=1}^{k-1} \log \frac{R_k(x_t)}{R_j(x_t)} \right]^{-1}$. This estimator is asymptotically unbiased (see [10] for more details). If the data points belong to the same manifold, we can average over all $m(x_t)$ in order to obtain a more robust estimator. However, if there are two or more manifolds with different dimensions, the average does not make sense, unless we first cluster according to dimensionality and then we estimate the dimensionality for each

cluster. We briefly toy with this idea now, as a warm up to our simultaneous soft clustering and estimation technique described in Section 3.

## 2.1 A two step clustering approach

As a first simple approach to detect and cluster mixed dimensionality (and/or densities), we can combine a local dimensionality estimator such as the one just described and a clustering technique. For the second step we use the Information Bottleneck (IB) [17], which is an elegant framework to eventually combine several local dimension estimators and other possible features such as density [6]. The IB is a technique that allows to cluster (compress) a variable according to another related variable. Let $X$ be the set of variables to be clustered and $S$ the relevance variable that gives some information about $X$. An example is the information that different words provide about documents of different topics. We call $\tilde{X}$ the clustered version of $X$. The optimal $\tilde{X}$ is the one that minimizes the functional $\mathcal{L}(p(\tilde{x}_t|x_t)) = I(\tilde{X}; X) - \beta I(\tilde{X}; S)$, where $I(\cdot; \cdot)$ denotes mutual information and $p(\cdot)$ the probability density function. There is a trade-off, controlled by $\beta$, between compressing the representation and preserving the meaningful information. In our context, we want to cluster the data according to the intrinsic dimensionality (and/or density). Then, our relevant variable $S$ will be the set of (quantized) estimated local intrinsic dimensions. For the joint distribution $p(x_t, s_i)$, $s_i \in S$, we use the histogram of local dimensions inside a ball of radius $R'$ around $x_t$,[2] computed by the LB technique.

Examples of this technique will be presented in the experimental Section. Instead of a two-steps algorithm, with local dimensionality and/or density estimation followed by clustering, we now propose a maximum likelihood technique that combines these steps.

## 3 Poisson mixture model

The core approach that we propose to study stratifications (mixed manifolds) is based on extending the LB technique [10]. Instead of modelling each point and its local ball of radius $R$ as a Poisson process and computing the ML for each ball separately, we consider all the possible balls at the same time in the same ML function. As the probability density function for all the point cloud we consider a mixture of Poisson distributions with different parameters (dimension and density). Thus, we allow the presence of different intrinsic dimensions and densities in the dataset. These are automatically computed while being used for soft clustering.

Let us denote by $J$ the number of different Poisson distributions considered in the mixture, each one with a (possibly) different dimension $m$ and density parameter $\theta$. We consider the vector set of parameters $\psi = \{\psi^j = (\pi^j, \theta^j, m^j); j = 1, \ldots, J\}$, where $\pi^j$ is the mixture coefficient for class $j$ (the proportion of distribution $j$ in the dataset), $\theta^j$ is its density parameter ($f^j = e^{\theta^j}$), and $m^j$ is its dimension. We denote by $p(\cdot)$ the probability density function and by $P(\cdot)$ the probability.

As in the LB approach, the observable event will be $y_t = N(R, x_t)$, the number of points inside the ball $B(R, x_t)$ of radius $R$ centered at point $x_t$. The total number of observations is $T'$ and $Y = \{y_t; t = 1, \ldots, T'\}$ is the observation sequence. If we consider every possible ball in the dataset then, $T'$ coincides with the total number of points $T$ in the point cloud. From now on, we will consider this case and $T' \equiv T$. The density function of the Poisson mixture model is given by

$$p(y_t|\psi) = \sum_{j=1}^{J} \pi^j p(y_t|\theta^j, m^j) = \sum_{j=1}^{J} \pi^j \exp\left(\int_0^R \log \lambda^j(r) \, dN(r, x_t)\right) \exp\left(-\int_0^R \lambda^j(r) dr\right),$$

where $\lambda^j(r) = e^{\theta^j} V(m^j) m^j r^{m^j - 1}$. Usually, problems involving a mixture of experts are solved by the Expectation Maximization (EM) algorithm [5]. In our context, there are two kinds of unknown parameters: The membership function of an expert (class), $\pi^j$, and the parameters of each expert, $m^j$ and $\theta^j$. The membership information is originally unknown, thereby making the parameter estimation for each class difficult. The EM algorithm computes its expected value (E-step) and then this value is used for the parameter estimation procedure (M-step). These two steps are iterated.

If $Y$ contains $T$ statistically independent variables, then the incomplete data log-likelihood is:

$$L(Y|\psi) = \log p(Y|\psi) = \log \prod_{t=1}^{T} p(y_t|\psi) = Q(\psi) + R(\psi),$$

$$Q(\psi) := \sum_{Z} P(Z|Y,\psi) \log p(Z,Y|\psi), \quad R(\psi) := -\sum_{Z} P(Z|Y,\psi) \log P(Z|Y,\psi),$$

where $Z = \{z_t \in C; t = 1, \ldots, T\}$ is the missing data (hidden-state information), and the set of class labels is $C = \{C^1, C^2, \ldots C^J\}$. Here, $z_t = C^j$ means that the $j$-th mixture generates $y_t$. We call $Q$ the expectation of $\log p(Z, Y|\psi)$ with respect to $Z$. The EM algorithm is based on maximizing $Q$, since while improving (maximizing) the function $Q$ at each iteration, the likelihood function $L$ is also improved. The probability density that appears in the function $Q$ can be written as $p(Z, Y|\psi) = \prod_{t=1}^{T} p(z_t, y_t|\psi)$, and the complete-data log-likelihood becomes

$$\log p(Z, Y|\psi) = \sum_{t=1}^{T} \sum_{j=1}^{J} \delta_t^j \log \left[ p(y_t|z_t = C^j, \psi^j)\pi^j \right], \tag{1}$$

where a set of indicator variables $\delta_t^j$ is used in order to indicate the status of the hidden variables:

$$\delta_t^j \equiv \delta(z_t, C^j) = \begin{cases} 1 & \text{if } y_t \text{ generated by mixture } C^j, \\ 0 & \text{else.} \end{cases}$$

Considering the expectation, with respect to $Z$, $E_Z(\cdot)$ of (1) and setting $\psi$ to a fixed known value $\psi_n$ (the value at step $n$ of the algorithm), everywhere except for the log function, we get a function $Q$ of $\psi$. We denote it by $Q(\psi|\psi_n)$, and it has the following form

$$Q(\psi|\psi_n) = \sum_{t=1}^{T} \sum_{j=1}^{J} h_n^j(y_t) \log \left[ p(y_t|\delta_t^j = 1, \psi^j)\pi^j \right],$$

where

$$h_n^j(y_t) = E_Z[\delta_t^j|y_t, \psi_n] = P(\delta_t^j = 1|y_t, \psi_n) = \frac{p(y_t|\delta_t^j = 1, \psi_n^j)\pi_n^j}{\sum_{l=1}^{J} p(y_t|\delta_t^l = 1, \psi_n^l)\pi_n^l} \tag{2}$$

is the probability that observation $t$ belongs to mixture $j$. Finally, the probability density in (2) is

$$p(y_t|\delta_t^j = 1, \psi_n^j) = \exp \left( \int_0^R \log \lambda_n^j(r) dN(r, x_t) \right) \exp \left( -\int_0^R \lambda_n^j(r) dr \right), \tag{3}$$

where $\lambda_n^j(r) = e^{\theta_n^j} V(m_n^j) m_n^j r^{m_n^j - 1}$. As mentioned above, the EM algorithm consists of two main steps. In the E-step, the function $Q(\psi|\psi_n)$ is computed, for that, we determine the best guess of the membership function, i.e., the probabilities $h_n^j(y_t)$. Once we know these probabilities, $Q(\psi|\psi_n)$ can be considered as a function of the only unknown, $\psi$, and it is maximized in order to compute the values of $\psi_{n+1}$, i.e., the maximum likelihood parameters $\psi$ at step $n + 1$; this is called the M-step. The EM suffers from local maxima, hitting a local maximum can be prevented running the algorithm several times with different initializations. Different random subset of points, from the point cloud, may be used in each run. We have experimented with both approaches and the results are always similar if we initialize all the probabilities equally. The Algorithm PMM describes the main components of this proposed approach. The estimators $\pi_{n+1}^j$, $m_{n+1}^j$, and $\theta_{n+1}^j$ are obtained by computing $\psi_{n+1}^j = \arg\max_{\psi^j} Q(\psi|\psi_n) + \lambda(\sum_{l=1}^{J} \pi^l - 1)$ in the M-step, where $\lambda$ is the Lagrange multiplier that allows to introduce the constraint $\sum_{l=1}^{J} \pi^l = 1$. This gives equations (4)-(5), where $V(m_n^j) = (2\pi^{m_n^j/2})/(m_n^j \Gamma(\frac{m_n^j}{2}))$, and $\Gamma(\frac{m_n^j}{2}) = \int_0^\infty t^{m_n^j/2 - 1} e^{-t} dt$. In order to compute $m_{n+1}^j$ we have used the same approach as in [10], by means of a $k$ nearest neighbor graph.

## 4 Experimental results

We now present a number of experimental results for the technique proposed in Section 3. We often compare it with the two-steps algorithm described in Section 2, and denote this algorithm by LD+IB.

**Algorithm PMM** *Poisson Mixture Model*

**Require:** The point cloud data, $J$ (number of desired classes) and $k$ (scale of observation).
**Ensure:** Soft clustering according to dimensionality and density.

1: Initialization of $\psi_0 = \{\pi_0^j, m_0^j, \theta_0^j\}$ to any set of values which ensures that $\sum_{j=1}^{J} \pi_0^j = 1$.
2: EM iterations on $n$,
    For all $j = 1, \ldots J$, compute:
        &bull; E-step: Compute $h_n^j(y_t)$ by (2).
        &bull; M-step: Compute

$$\pi_{n+1}^j = \frac{1}{T}\sum_{t=1}^{T} h_n^j(y_t); \quad m_{n+1}^j = \left[\frac{\sum_{t=1}^{T} h_n^j(y_t)\sum_{j=1}^{k-1}\log\frac{R_k(y_t)}{R_j(y_t)}}{\sum_{t=1}^{T} h_n^j(y_t)(k-1)}\right]^{-1} \quad (4)$$

$$\theta_{n+1}^j = \log\sum_{t=1}^{T} h_n^j(y_t)(k-1) - \log\left(V(m_n^j)\sum_{t=1}^{T} h_n^j(y_t)R_k(y_t)^{m_n^j}\right) \quad (5)$$

Until convergence of $\psi_n$, that is, when $\|\psi_{n+1} - \psi_n\|_2 < \epsilon$, for a certain small value $\epsilon$.

In all the experiments we use the initialization $\pi_0^j = 1/J$, $\theta_0^j = 0$, and $m_0^j = j$, for all $j = 1, \ldots, J$. The distances are normalized so that the maximum distance is 1. The embedding dimension in all the experiments on synthetic data is 3, although the results were found to be consistent when we increased the embedding dimension.

The first experiment consists of a mixture of a Swiss roll manifold (700 points) and a line (700 points) embedded in a three dimensional space. The algorithm (with $J = 2$ and $k = 10$) is able to separate both manifolds. The estimated parameters are collected in Table 1. For each table, we display the estimated dimension $m$, density $\theta$, and mixture coefficient $\pi$ for each one of the classes. We also show the percentage of points of each manifold that are classified in each class (after thresholding the soft assignment). Figure 1(a) displays both manifolds – each point is colored according to the probability of belonging to each one of the two possible classes. Tables 1(a) and 1(c) contain the results for both PMM and LD+IB using $J = 2$. Table 1(b) shows the results for the PMM algorithm with $k = 10$ and $J = 3$. Note how the parameters of the first two classes are quite similar to the ones obtained with $J = 2$, and the third class is marginal (very small $\pi$). Figure 1(b) shows the PMM classification when $J = 3$. Note that all the points of the line belong to the class of dimension 1. The points of the Swiss roll are mainly concentrated in the other class with dimension 2. A slight amount of Swiss roll points belong to a third class with roughly the same dimension as the second class. Actually, these points are located in the point cloud boundaries, where the underlying assumptions are not always valid.

If we estimate the dimension of the mixture using the LB technique with $k = 10$, we obtain 1.70 with a standard deviation of 5.31. If we use the method proposed by Costa and Hero [4], the estimated dimension is 2. In both cases, the estimated intrinsic dimension is the largest one present in the mixture, ignoring that the data actually lives in two manifolds of different intrinsic dimension.

The same Table and Figure, second rows, show results for noisy data. We add to the point coordinates Gaussian noise with $\sigma = 0.6$. The results obtained with $k = 10$ are displayed in Tables 1(d), 1(e) and 1(f), and in Figures 1(d), 1(e) and 1(f). Note how the classification still separates the two different manifolds, although the line is much more affected by the noise and it does not look like a one dimensional manifold anymore. This is reflected also by the estimated dimension which now is bigger. This phenomena is related to the scale of observation and to the level of noise. If the level of noise is large – e.g., compared to the mean distance to the $k$ nearest neighbors for a small $k$ – intuitively the estimated intrinsic dimension will be closer to the embedding dimension (this behavior was experimentally verified). We can again compare the results with the ones obtained with the LB estimator alone: Estimated dimension 2.71 and standard deviation 1.12. Using Costa and Hero [4], the estimated dimension varies between 2 and 3 (depending on the number of bootstrap loops). Both techniques do not consider the possibility of mixed dimensionality.

The experiment in Figure 2 illustrates how the soft clustering is done according to both dimensionality and density. The data consists of 2500 points on the Swiss roll, 100 on a line with high density

| Estimated parameters | | |
|---|---|---|
| $m$ | 1.00 | 2.01 |
| $\theta$ | 5.70 | 2.48 |
| $\pi$ | 0.5000 | 0.5000 |
| % points in each class | | |
| Line | 100 | 0 |
| SR | 0 | 100 |

(a) PMM ($J = 2$).

| Estimated parameters | | | |
|---|---|---|---|
| $m$ | 1.00 | 2.01 | 2.16 |
| $\theta$ | 5.70 | 2.55 | 1.52 |
| $\pi$ | 0.5000 | 0.4792 | 0.0208 |
| % points in each class | | | |
| Line | 100 | 0 | 0 |
| SR | 0 | 96.57 | 3.43 |

(b) PMM ($J = 3$).

| Estimated dimension | | |
|---|---|---|
| $m$ | 1.67 | 2.00 |
| % points in each class | | |
| Line | 100 | 0 |
| Swiss roll | 3.45 | 96.55 |

(c) LD+IB ($J = 2$).

| Estimated parameters | | |
|---|---|---|
| $m$ | 3.02 | 2.38 |
| $\theta$ | 7.69 | 2.73 |
| $\pi$ | 0.4951 | 0.5049 |
| % points in each class | | |
| Line | 98.14 | 1.86 |
| SR | 0.86 | 99.14 |

(d) PMM ($J = 2$).

| Estimated parameters | | | |
|---|---|---|---|
| $m$ | 3.01 | 2.40 | 2.26 |
| $\theta$ | 7.70 | 2.88 | 1.72 |
| $\pi$ | 0.4910 | 0.4766 | 0.0325 |
| % points in each class | | | |
| Line | 97.71 | 2.29 | 0 |
| SR | 0.71 | 93.00 | 6.29 |

(e) PMM ($J = 3$).

| Estimated dimension | | |
|---|---|---|
| $m$ | 3.09 | 2.30 |
| % points in each class | | |
| Line | 79.71 | 20.29 |
| Swiss roll | 24.71 | 75.29 |

(f) LD+IB ($J = 2$).

Table 1: *Clustering results for the Swiss roll (SR) and a line ($k = 10$), without noise (first row) and with noise (second row).*

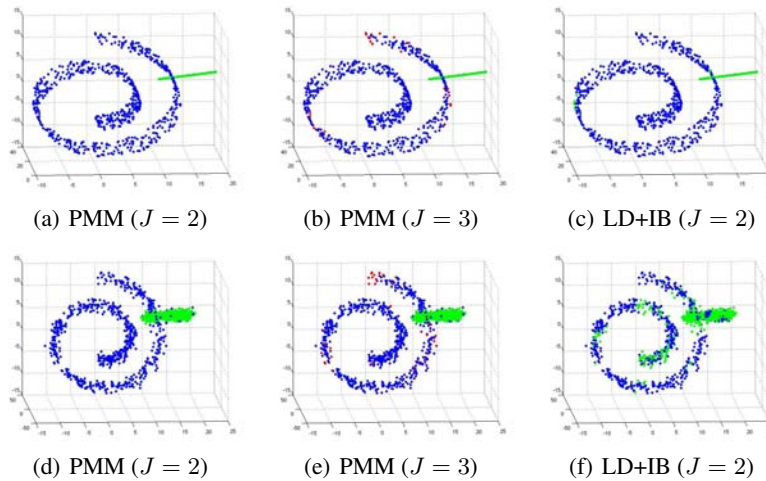

(a) PMM ($J = 2$)   (b) PMM ($J = 3$)   (c) LD+IB ($J = 2$)

(d) PMM ($J = 2$)   (e) PMM ($J = 3$)   (f) LD+IB ($J = 2$)

Figure 1: *Clustering of a line and a Swiss roll ($k = 10$). First row without noise, second row with Gaussian noise ($\sigma = 0.6$). Points colored according to the probability of belonging to each class.*

and 50 on another less dense line. We have set $J = 4$ and the algorithm gives an "empty class," thus discovering that three classes, with correct dimensionality and density, is enough for a good representation. The only errors are in the borders, as expected.

| Estimated parameters | | | | |
|---|---|---|---|---|
| $m$ | 1.94 | 1.04 | 0.98 | 1.93 |
| $\theta$ | 7.12 | 3.82 | 2.66 | 2.57 |
| $\pi$ | 0.9330 | 0.0498 | 0.0167 | 0.0004 |
| % points in each class | | | | |
| Line | 0.0 | 15.69 | 84.31 | 0.0 |
| Line (dense) | 0.0 | 99.00 | 1.00 | 0.0 |
| Swiss Roll | 98.92 | 1.08 | 0.0 | 0.0 |

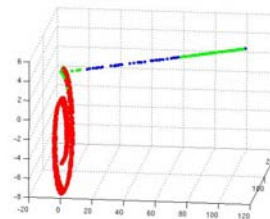

Figure 2: *Clustering with mixed dimensions and density ($k = 20$, $J = 4$).*

In order to test the algorithm with real data, we first work with the MNIST database of handwritten digits,[3] which has a test set of 10.000 examples. Each digit is an image of $28 \times 28$ pixels and we treat the data as 784-dimensional vectors.

We study the mixture of digits *one* and *two* and apply PMM and LD+IB with $J = 2$ and $k = 10$. The results are shown in Figure 3. Note how the digits are well separated.[4] The LB estimator alone gives dimensions 9.13 for digits *one*, 13.02 for digits *two*, and 11.26 for the mixture of both digits. The Costa and Hero's method, [4], gives 8, 11 and 9 respectively. Both methods assume a single intrinsic dimension and give an average of the dimensions of the underlying manifolds.

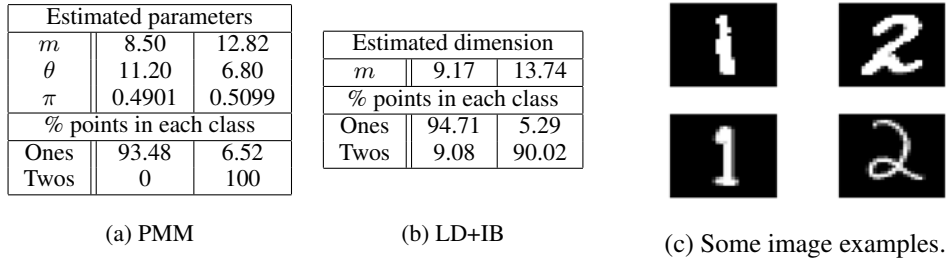

| Estimated parameters | | |
|---|---|---|
| $m$ | 8.50 | 12.82 |
| $\theta$ | 11.20 | 6.80 |
| $\pi$ | 0.4901 | 0.5099 |
| % points in each class | | |
| Ones | 93.48 | 6.52 |
| Twos | 0 | 100 |

| Estimated dimension | | |
|---|---|---|
| $m$ | 9.17 | 13.74 |
| % points in each class | | |
| Ones | 94.71 | 5.29 |
| Twos | 9.08 | 90.02 |

(a) PMM
(b) LD+IB
(c) Some image examples.

Figure 3: *Results for digits 1 and 2 ($k = 10$, $J = 2$).*

Next, we experiment with 9-dimensional vectors formed of image patches of $3 \times 3$ pixels. If we impose $J = 3$ and use PMM, we obtain the results in Figure 4. Notice how roughly one class corresponds to patches in homogeneous zones (approximately constant gray value), a second class corresponds to textured zones and a third class to patches containing edges. The estimated dimensions in each region are in accordance to the estimated dimensions using Isomap or Costa and Hero's technique in each region after separation. This experiment is just a proof of concept, in the future we will study how to adapt this clustering approach to image segmentation.

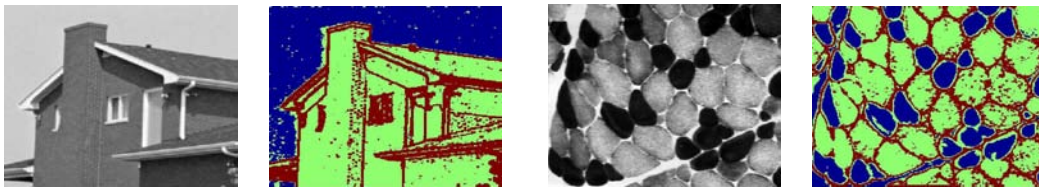

Figure 4: *Clustering of image patches of $3 \times 3$ pixels with PMM, colors indicating the different classes (complexity) ($J = 3$, $k = 30$). Left: original and segmented images of a house. Right: original and segmented images of a portion of biological tissue. Adding spatial regularization is the subject of current research.*

Finally, as an additional proof of the validity of our approach and its potential applications, we use the PMM framework to separate activities in video, Figure 5 (see also [14]). Each original frame is $480 \times 640$, sub-sampled to $48 \times 64$ pixels, with 1673 frames. Four classes are present: standing, walking, jumping, and arms waving. The whole run took 361 seconds in Matlab, while the classification time (PMM) can be neglected compared to the kNN component.

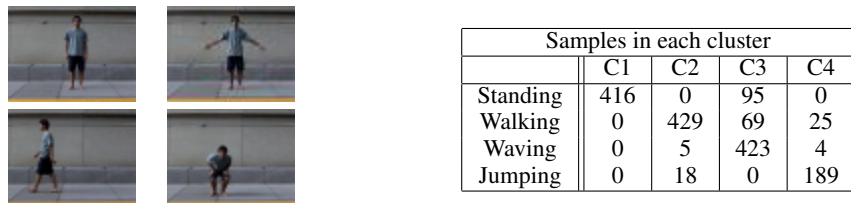

| Samples in each cluster | | | | |
|---|---|---|---|---|
| | C1 | C2 | C3 | C4 |
| Standing | 416 | 0 | 95 | 0 |
| Walking | 0 | 429 | 69 | 25 |
| Waving | 0 | 5 | 423 | 4 |
| Jumping | 0 | 18 | 0 | 189 |

Figure 5: *Classifying human activities in video ($k = 10$, $J = 4$). Four sample frames are shown followed by the classification results (confusion matrix). Visual analysis of the wrongly classified frames show that these are indeed very similar to the misclassified class members. Adding features, e.g., optical flow, will improve the results.*

## 5 Conclusions

In this paper we discussed the concept of "stratification learning," where the point cloud data is not assumed to belong to a single manifold, as commonly done in manifold learning and dimensionality reduction. We extended the work in [10] in the sense that the maximum likelihood is computed once for the whole dataset, and the probability density function is a mixture of Poisson laws, each one modeling different intrinsic dimensions and densities. The soft clustering and the estimation are simultaneously computed. This framework has been contrasted with a more standard two-steps approach, a combination of the local estimator introduced in [10] with the Information Bottleneck clustering technique [17]. In both methods we need to compute a kNN-graph which is precisely the computationally more expensive part. The mixture of Poisson estimators is faster than the two-steps approach one, it uses an EM algorithm, linear in the number of classes and observations, which converges in a few iterations.

The mixture of Poisson model is not only clustering according to dimensionality, but to density as well. The introduction of additional observations and estimates can also help to separate points that although have the same dimensionality and density, belong to different manifolds. We would also like to study the use of ellipsoids instead of balls in the counting process in order to better follow the geometry of the intrinsic manifolds. Another aspect to study is the use of metrics more adapted to the nature of the data instead of the Euclidean distance. At the theoretical level, the bias of the PMM model needs to be studied. Results in these directions will be reported elsewhere.

**Acknowledgments:** This work has been supported by ONR, DARPA, NSF, NGA, and the McKnight Foundation. We thank Prof. Persi Diaconis and Prof. René Vidal for important feedback and comments. We also thank Pablo Arias and Jérémie Jakubowicz for their help. GR was on sabbatical from the Universidad de la Republica, Uruguay, while performing this work.

## Footnotes

[1]M. Hein pointed us out in NIPS that this dimension estimator is equivalent to the one proposed in [15].

[2]The value of $R'$ determines the amount of regularity in the classification.

[3]http://yann.lecun.com/exdb/mnist/

[4]Since the clustering is done according to dimensionality and density, digits which share these characteristics won't be separated into different classes.

## References

[1] D. Barbara and P. Chen. Using the fractal dimension to cluster datasets. In *Proceedings of the Sixth ACM SIGKDD*, pages 260–264, 2000.

[2] M. Belkin and P. Niyogi. Laplacian eigenmaps and spectral techniques for embedding and clustering. In *Advances in NIPS 14*, 2002.

[3] M. Brand. Charting a manifold. In *Advances in NIPS 16*, 2002.

[4] J. A. Costa and A. O. Hero. Geodesic entropic graphs for dimension and entropy estimation in manifold learning. *IEEE Trans. on Signal Processing*, 52(8):2210–2221, 2004.

[5] A. Dempster, N. Laird, and D. Rubin. Maximum likelihood from incomplete data. *Journal of the Royal Statistical Society Ser. B*, 39:1–38, 1977.

[6] N. Friedman, O. Mosenzon, N. Slonim, and N. Tishby. Multivariate information bottleneck. In *Seventeenth Conference UAI*, pages 152–161, 2001.

[7] A. Gionis, A. Hinneburg, S. Papadimitriu, and P. Tsparas. Dimension induced clustering. In *Proceeding of the Eleventh ACM SIGKDD*, pages 51–60, 2005.

[8] K. Huang, Y. Ma, and R. Vidal. Minimum effective dimension for mixtures of subspaces: A robust GPCA algorithm and its applications. In *Proceedings of CVPR*, pages 631–638, 2004.

[9] B. Kegl. Intrinsic dimension estimation using packing numbers. In *Advances in NIPS 14*, 2002.

[10] E. Levina and P.J. Bickel. Maximum likelihood estimation of intrinsic dimension. In *Advances in NIPS 17*, 2005.

[11] P. Mordohai and G. Medioni. Unsupervised dimensionality estimation and manifold learning in high-dimensional spaces by tensor voting. In *IJCAI*, page 798, 2005.

[12] S. T. Roweis and L. K. Saul. Nonlinear dimensionality reduction by locally linear embedding. *Science*, 290(5500):2323–2326, 2000.

[13] D. L. Snyder. *Random Point Processes*. Wiley, New York, 1975.

[14] R. Souvenir and R. Pless. Manifold clustering. In *ICCV*, pages 648–653, 2005.

[15] F. Takens. On the numerical determination of the dimension of an attractor. *Lecture notes in mathematics. Dynamical systems and bifurcations*, 1125:99–106, 1985.

[16] J. B. Tenenbaum, V. de Silva, and J. C. Langford. A global geometric framework for nonlinear dimensionality reduction. *Science*, 290(5500):2319–2323, 2000.

[17] N. Tishby, F. Pereira, and W. Bialek. The information bottleneck method. In *Proceedings of the 37-th Annual Allerton Conference on Communication, Control and Computing*, pages 368–377, 1999.